# Implementing *Intelligence* on Silicon Using Neuron–Like Functional MOS Transistors

Tadashi Shibata, Koji Kotani, Takeo Yamashita, Hiroshi Ishii
Hideo Kosaka, and Tadahiro Ohmi
Department of Electronic Engineering
Tohoku University
Aza–Aoba, Aramaki, Aobaku, Sendai 980 JAPAN

## Abstract

We will present the implementation of intelligent electronic circuits realized for the first time using a new functional device called *Neuron MOS Transistor* (neuMOS or νMOS in short) simulating the behavior of biological neurons at a single transistor level. Search for the most resembling data in the memory cell array, for instance, can be automatically carried out on hardware without any software manipulation. *Soft Hardware*, which we named, can arbitrarily change its logic function in real time by external control signals without any hardware modification. Implementation of a neural network equipped with an on–chip self–learning capability is also described. Through the studies of νMOS intelligent circuit implementation, we noticed an interesting similarity in the architectures of νMOS logic circuitry and biological systems.

## 1 INTRODUCTION

The motivation of this work has stemmed from the invention of a new functional transistor which simulates the behavior of biological neurons (Shibata and Ohmi, 1991; 1992a). The transistor can perform weighted summation of multiple input signals and squashing on the sum all at a single transistor level. Due to its functional similarity, the transistor was named Neuron MOSFET (abbreviated as neuMOS or νMOS). What is of significance with this new device is that a number of new architecture electronic circuits can be build using νMOS' which are different from conventional ones both in operational principles and functional capabilities. They are characterized by a high degree of parallelism in hardware computation, a large flexibility in hardware configuration and a dramatic reduction in the circuit complexity as compared to conventional integrated

circuits. During the course of studies in exploring νMOS circuit applications an interesting similarity has been noticed between the basic νMOS logic circuit architecture and the common structure found in biological neuronal systems, i. e., the competitive processes of excitatory and inhibitory connections. The purpose of this article is to demonstrate how powerful the neuron–like functionality in an elemental device is in implementing intelligent functions in silicon integrated circuits.

## 2 NEURON MOSFET AND *SOFT-HARDWARE LOGIC* CIRCUITS

The symbolic representation of a νMOS is given in Fig. 1. A νMOS is a regular MOS transistor except that its gate electrode is made electrically floating and multiple input terminals are capacitively coupled to the floating gate. The potential of the floating gate $\phi_F$ is determined as a linear weighted sum of multiple input voltages where each weighting factor is given by the magnitude of a coupling capacitance. When $\phi_F$, the weighted sum, exceeds the threshold voltage of the transistor, it turns on. Thus the function of a neuron model (McCulloch and Pitts, 1943) has been directly implemented in a simple transistor structure. νMOS transistors were fabricated using the double-polysilicon gate technology and a CMOS process was employed for νMOS integrated circuits fabrication. It should be noted here that no floating–gate charging effect was employed in the operation of νMOS logic circuits.

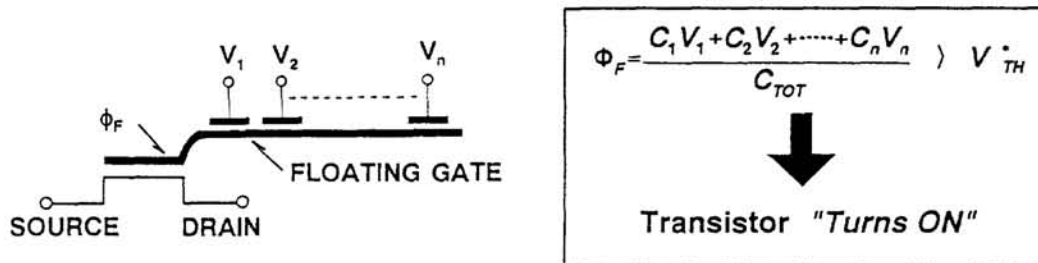

Figure 1: Schematic of a neuron MOS transistor.

Since the weighting factors in a νMOS are determined by the overlapping areas of the first poly (floating gate) and second poly (input gate) patterns, they are not alterable. For this reason, in νMOS applications to self–learning neural network synthesis, a synapse cell circuit was provided to each input terminal of a νMOS to represent an alterable connection strength. Here the plasticity of a synaptic weight was created by charging/discharging of the floating–gate in a νMOS synapse circuitry as described in **4**.

The *I–V* characteristics of a two–input–gate νMOS having identical coupling capacitances are shown in Fig. 2, where one of the input gates is used as a gate terminal and the other as a threshold–control terminal. The apparent threshold voltage as seen from the gate terminal is changed from a depletion–mode to an enhancement–mode threshold by the voltage given to the control terminal. This variable threshold nature of a νMOS, we believe, is most essential in creating flexibility in electronic hardware systems.

Figure 3(a) shows a two–input–variable *Soft Hardware Logic (SHL)* circuit which can represent all possible sixteen Boolean functions for two binary inputs $X_1$ and $X_2$ by adjusting the control signals $V_A$, $V_B$ and $V_C$. The inputs, $X_1$ and $X_2$, are directly coupled to the floating gate of a complementary νMOS inverter in the output stage with a 1:2 coupling ratio. The νMOS inverter, which we call the main inverter, determines the logic.

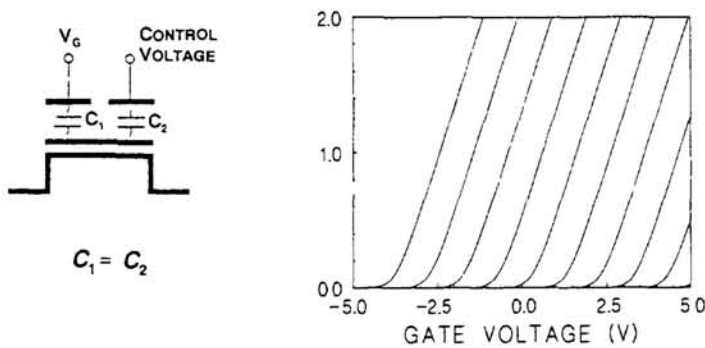

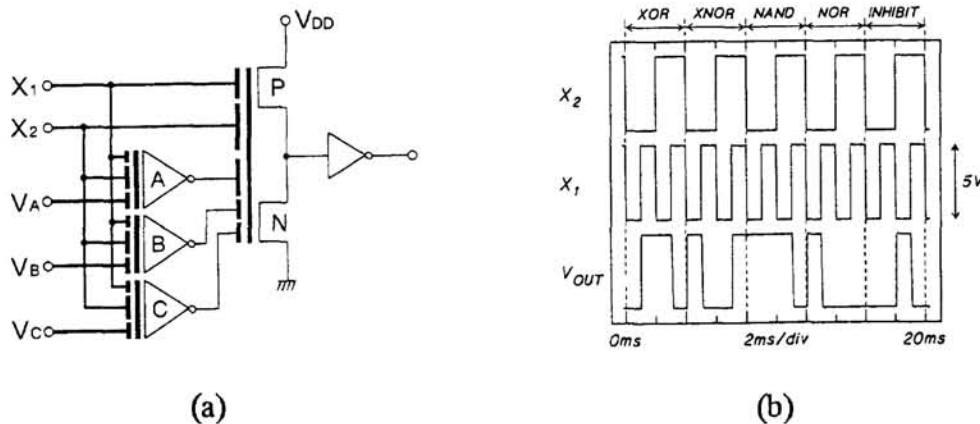

Figure 2: Measured characteristics of a variable threshold transistor. Voltage at the threshold–control terminal was varied from +5V to –5V (from left to right).

(a)                                    (b)

Figure 3:    Two–input–variable soft hardware logic circuit(a) and measured characteristics(b). The slow operation is due to the loading effect. (The test circuit has no output buffers.)

The inputs are also coupled to the main inverter via three inter–stage vMOS inverters (pre–inverters). When the analog variable represented by the binary inputs $X_1$ and $X_2$ increases, the inputs tend to turn on the main inverter via direct connection, while the indirect connection via pre–inverters tend to turn off the main inverter because pre–inverter outputs change from $V_{DD}$ to 0 when they turn on. This competitive process creates logics. The turn–on thresholds of pre–inverters are made alterable by control signals utilizing the variable threshold characteristics of vMOS'. Thus the real–time alteration of logic functions has been achieved and are demonstrated by experiments in Fig. 3(b). With the basic circuit architecture of the two–staged vMOS inverter configuration shown in Fig. 3(a), any Boolean function can be generated. We found the inverting connections via pre–inverters are most essential in logic synthesis. The structure indicates an interesting similarity to neuronal functional modules in which intramodular inhibitory connections play essential roles.

Fixed function logics can be generated much more simply using the basic two–staged structure, resulting in a dramatic reduction in transistor counts and interconnections. It has been demonstrated that a full adder, 3–b and 4–b A/D converters can be constructed only with 8, 16 and 28 transistors, respectively, which should be compared to 30, 174 and 398 transistors by conventional CMOS design, respectively. The details on vMOS circuit design are described in Refs. (Shibata and Ohmi, 1993a; 1993b) and experimental verification in Ref. (Kotani et al., 1992).

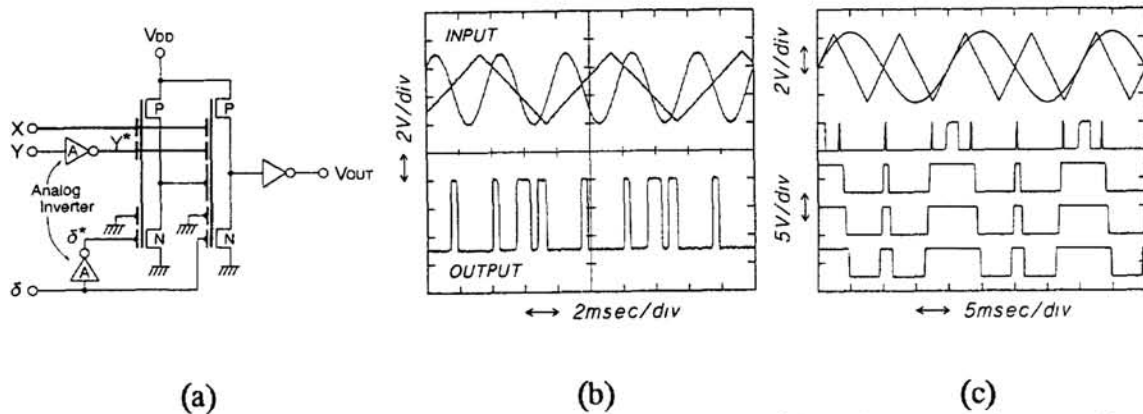

(a)                          (b)                          (c)

Figure 4: Real–time rule–variable data matching circuit (a) and measured wave forms (b & c). In (c), δ is changed as 0.5, 1, 1.5, and 2 [V] from top to bottom.

A unique νMOS circuit based on the basic structure of Fig. 3(a) is the real–time rule–variable data matching circuitry shown in Fig. 4(a). The circuit output becomes high when $|X - Y| < \delta$. X is the input data, Y the template data and δ the window width for data matching where X, Y and δ are all time variables. Measured data are shown in Figs. 4(b) and 4(c), where it is seen the triple peaks are merged into a single peak as δ increases (Shibata et al., 1993c). The circuit is composed of only 10 νMOS' and can be easily integrated with each pixel on a image sensor chip. If νMOS circuitry is combined with a bipolar image sensor cell having an amplification function (Tanaka et al., 1989), for instance, *in situ* image processing such as edge detection and variable–template matching would become possible, leading to an intelligent image sensor chip.

## 3  BINARY–MULTIVALUED–ANALOG MERGED HARDWARE COMPUTATION

A winner–take–all circuit (WTA) implemented by νMOS circuitry is given in Fig. 5. Each cell is composed of a νMOS variable threshold inverter in which the apparent threshold is modified by an analog input signals $V_A \sim V_C$. When the common input signal $V_R$ is ramped up, the lowest threshold cell (a cell receiving the largest analog input) turns on firstly, at which instance a feedback loop is formed in each cell and the state of the cell is self–latched. As a result, only the winner cell yields an output of 1. The circuit has been applied to building an associative memory as demonstrated in Fig. 6. The binary data stored in a SRAM cell array are all simultaneously matched to the sample data by taking XNOR, and the number of matched bits are transferred to the floating gate of each WTA cell by capacitance coupling. The WTA action finds the location of data having the largest number of matched bits. This principle has been also applied to an sorting circuitry (Yamashita et al., 1993). In these circuits all computations are conducted by an algorithm directly imbedded in the hardware. Such an analog–digital merged hardware computation algorithm is a key to implement intelligent data processing architecture on silicon. A multivalued DRAM cell equipped with the association function and a multivalued SRAM cell having self–quantizing and self–classification functions have been also developed based on the binary–multivalued–analog merged hardware algorithm (Rita et al., 1994).

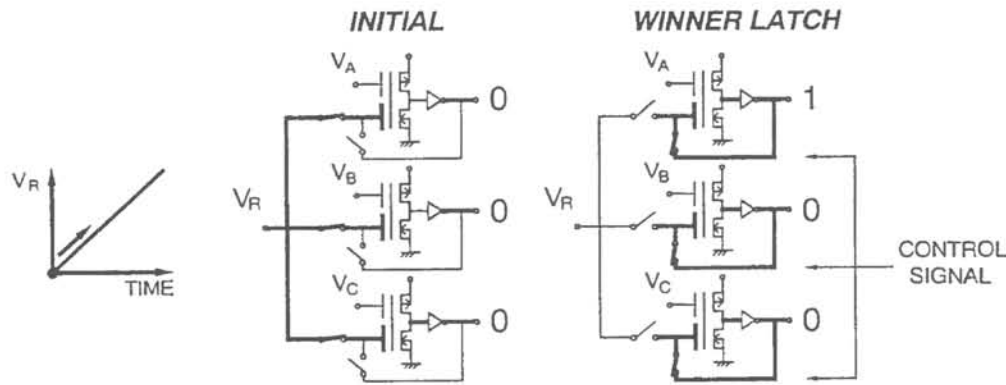

Figure 5: Operational principle of vMOS Winner–Take–All circuit.

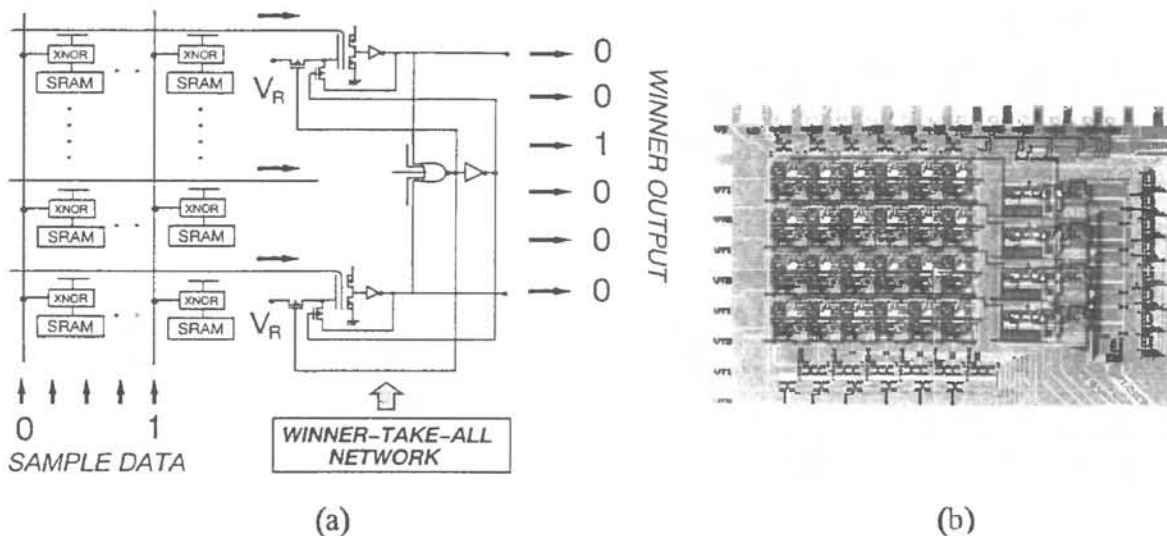

Figure 6: vMOS associative memory: (a) circuit diagram; (b) photomicrograph of a test chip.

## 4  HARDWARE SELF–LEARNING NEURAL NETWORKS

Since vMOS itself has the basic function of a neuron, a neuron cell is very easily implemented by a complementary vMOS inverter. The learning capability of a neural network is due to the plasticity of synaptic connections. Therefore its circuit implementation is a key issue. A stand–by power dissipation free synapse circuit which has been developed using vMOS circuitry is shown in Fig. 7(a). The circuit is a differential pair of N–channel and P–channel vMOS source followers sharing the same floating gate, which are both merged into CMOS inverters to cut off dc current paths. When the pre–synaptic neuron fires, both source followers are activated. Then the analog weight value stored as charges in the common floating gate is read out and transferred to the floating gate (dendrite) of the post–synaptic neuron by capacitance coupling as shown in Figs. 7 (b) and (c). The outputs of N–vMOS ($V^+$) and P–vMOS ($V^-$) source followers are averaged at the dendrite level, yielding an effective synapse output equal to $(V^+ + V^-)/2$. The synapse can represent both positive (excitatory) and negative (inhibitory) weights depending on whether the effective output is larger or smaller than $V_{DD}/2$, respectively. The operation of the synapse cell is demonstrated in Fig. 8(a).

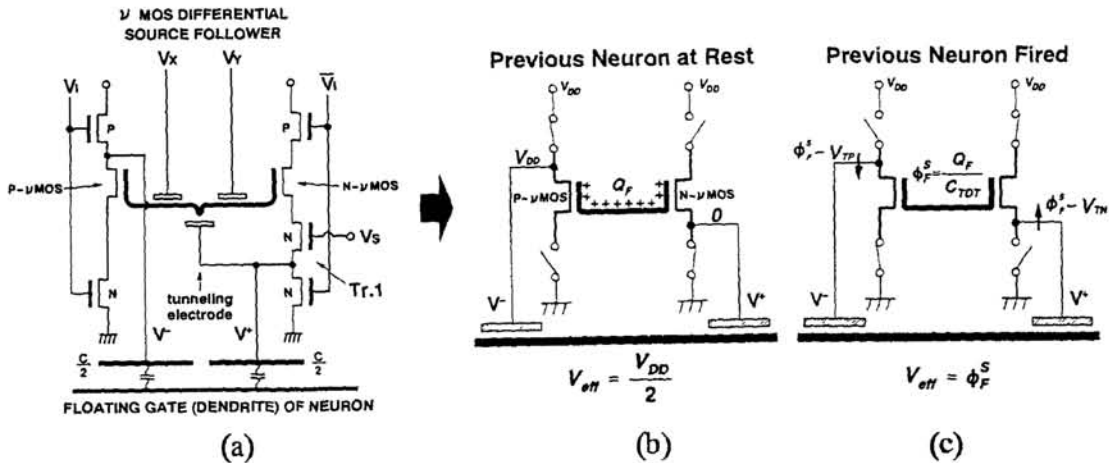

Figure 7: Synapse cell circuit implemented by νMOS circuitry.

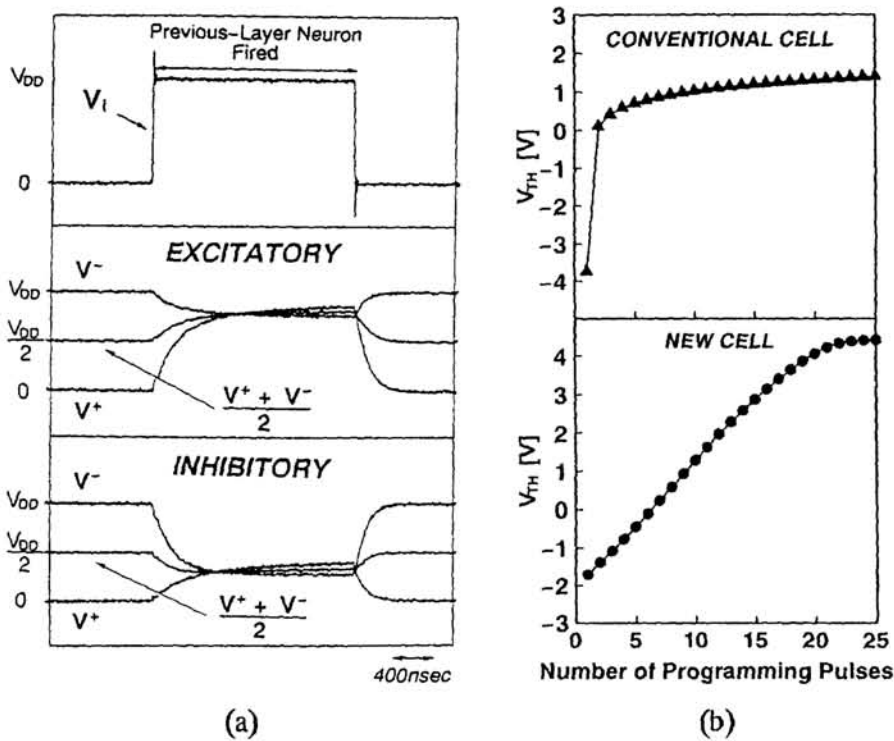

Figure 8: (a) Measured synapse cell output characteristics; (b) weight updating characteristics as represented by N–νMOS threshold with (our new cell:bottom) or without (conventional EEPROM cell: top) feed back.

The weight updating is conducted by giving high programming pulses to both $V_X$ and $V_Y$ terminals. (Their coupling capacitances are made much larger than others). Then the common floating gate is pulled up to the programming voltage, allowing electrons to flow into the floating gate via Fowler–Nordheim tunneling. When either $V_X$ or $V_Y$ is low, tunneling injection does not occur because the tunneling current is very sensitive to the electric field intensity, being exponentially dependent upon the tunnel oxide field (Hieda

et al., 1985). The data updating occurs only at the crossing point of $V_X$ and $V_Y$ lines, allowing Hebb–rule–like learning directly implemented on the hardware (Shibata and Ohmi, 1992b). Hardware–Backpropagation (HBP) learning algorithm, which is a simplified version of the original BP, has been also developed in order to facilitate its hardware implementation (Ishii et al., 1992) and has been applied to build self–learning vMOS neural networks (Ishii et al., 1993).

One of the drawbacks of programming by tunneling is the non–linearity in the data updating characteristics under constant pulses as shown in Fig. 8(b) (top). This difficulty has been beautifully resolved in our cell. With $V_S$ high, the output of the N–vMOS source follower is fed back to the tunneling electrode and the floating–gate potential is set to the tunneling electrode. In this manner, the voltage across the tunneling oxide is always preset to a constant voltage (equal to the N–vMOS threshold) before a programming pulse is applied, thus allowing constant charge to be injected or extracted at each pulse (Kosaka et al, 1993) as demonstrated in Fig. 8(b) (bottom). A test self–learning circuit that learned XOR is shown in Fig. 9.

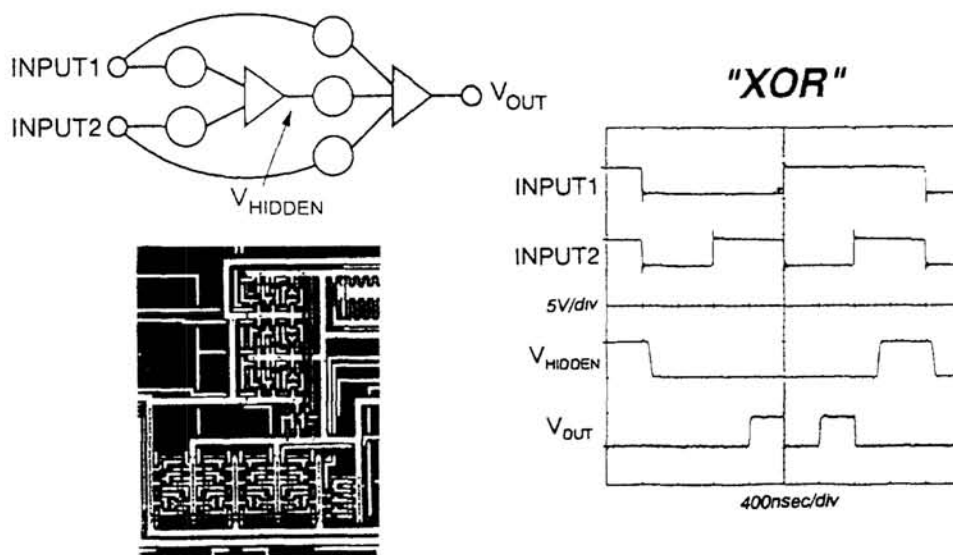

Figure 9:  Test circuit of vMOS neural network and its response when XOR is learnt.

## 5   SUMMARY

Development of intelligent electronic circuit systems using a new functional device called *Neuron MOS Transistor* has been described. vMOS circuitry is characterized by its high parallelism in computation scheme and the large flexibility in altering hardware functions and also by its great simplicity in the circuit organization. The ideas of *Soft Hardware* and the vMOS associative memory were not directly inspired from biological systems. However, an interesting similarity is found in their basic structures. It is also demonstrated that the vMOS circuitry is very powerful in building neural networks in which learning algorithms are imbedded in the hardware. We conclude that the neuron–like functionality at an elementary device level is essentially important in implementing sophisticated information processing algorithms directly in the hardware.

## ACKNOWLEDGMENT

This work was partially supported by the Grant–in–Aid for Scientific Research (04402029) and Grant–in–Aid for Developmental Scientific Research (05505003) from the Ministry of Education, Science and Culture, Japan. A part of this work was carried out in the Super Clean Room of Laboratory for Microelectronics, Research Institute of Electrical Communication, Tohoku University.

## REFERENCES

[1] T. Shibata and T. Ohmi, "An intelligent MOS transistor featuring gate–level weighted sum and threshold operations," in IEDM Tech. Dig., 1991, pp. 919–922.

[2] T. Shibata and T. Ohmi, "A functional MOS transistor featuring gate–level weighted sum and threshold operations," IEEE Trans. Electron Devices, Vol. 39, No. 6, pp.1444–1455 (1992a).

[3] W. S. McCulloch and W. Pitts, "A logical calculus of the ideas immanent in nervous activity," Bull. Math. Biophys., Vol. 5, pp. 115–133, 1943.

[4] T. Shibata and T. Onmi, "Neuron MOS binary–logic integrated circuits: Part I, Design fundamentals and soft–hardware–logic circuit implementation," IEEE Trans. Electron Devices, Vol. 40, No. 3, pp. 570–576 (1993a).

[5] T. Shibata and T. Ohmi, "Neuron MOS binary–logic integrated circuits: Part II, Simplifying techniques of circuit configuration and their practical applications," IEEE Trans. Electron Devices, Vol. 40, No. 5, 974–979 (1993b).

[6] K. Kotani, T. Shibata, and T. Ohmi, "Neuron–MOS binary–logic circuits featuring dramatic reduction in transistor count and interconnections," in IEDM Tech. Dig., 1992, pp. 431–434.

[7] T. Shibata, K. Kotani, and T. Ohmi, "Real–time reconfigurable logic circuits using neuron MOS transistors," in ISSCC Dig. Technical papers, 1993c, FA 15.3, pp. 238–239.

[8] N. Tanaka, T. Ohmi, and Y. Nakamura, "A novel bipolar imaging device with self–noise reduction capability," IEEE Trans. Electron Devices, Vol. 36, No. 1, pp. 31–38 (1989).

[9] T. Yamashita, T. Shibata, and T. Ohmi, "Neuron MOS winner–take–all circuit and its application to associative memory," in ISSCC Dig. Technical papers, 1993, FA 15.2, pp. 236–237.

[10] R. Au, T. Yamashita, T. Shibata, and T. Ohmi, "Neuron–MOS multiple–valued memory technology for intelligent data processing," in ISSCC Dig. Technical papers, 1994, FA 16.3.

[11] K. Hieda, M. Wada, T. Shibata, and H. Iizuka, "Optimum design of dual–control gate cell for high–density EEPROM's," IEEE Trans. Electron Devices, vol. ED–32, no. 9, pp. 1776–1780, 1985.

[12] T. Shibata and T. Ohmi, "A self–learning neural–network LSI using neuron MOSFET's," in Dig. Tech. Papers, 1992 Symposium on VLSI Technology, Seattle, June, 1992, pp. 84–85.

[13] H. Ishii, T. Shibata, H. Kosaka, and T. Ohmi, "Hardware–Backpropagation learning of neuron MOS neural networks," in IEDM Tech. Dig., 1992, pp. 435–438.

[14] H. Ishii, T. Shibata, H. Kosaka, and T. Ohmi, "Hardware–learning neural network LSI using a highly functional transistor simulating neuron actions," in Proc. International Joint Conference on Neural Networks '93, Nagoya, Oct. 25–29, 1993, pp. 907–910.

[15] H. Kosaka, T. Shibata, H. Ishii, and T. Ohmi, "An excellent weight–updating–linearity synapse memory cell for self–learning neuron MOS neural networks," in IEDM Tech. Dig., 1993, pp. 626–626.